# Learning to Agglomerate Superpixel Hierarchies

**Viren Jain**
Janelia Farm Research Campus
Howard Hughes Medical Institute

**Srinivas C. Turaga**
Brain & Cognitive Sciences
Massachusetts Institute of Technology

**Kevin L. Briggman, Moritz N. Helmstaedter, Winfried Denk**
Department of Biomedical Optics
Max Planck Institute for Medical Research

**H. Sebastian Seung**
Howard Hughes Medical Institute
Massachusetts Institute of Technology

## Abstract

An agglomerative clustering algorithm merges the most similar pair of clusters at every iteration. The function that evaluates similarity is traditionally hand-designed, but there has been recent interest in supervised or semisupervised settings in which ground-truth clustered data is available for training. Here we show how to train a similarity function by regarding it as the action-value function of a reinforcement learning problem. We apply this general method to segment images by clustering superpixels, an application that we call Learning to Agglomerate Superpixel Hierarchies (LASH). When applied to a challenging dataset of brain images from serial electron microscopy, LASH dramatically improved segmentation accuracy when clustering supervoxels generated by state of the boundary detection algorithms. The naive strategy of directly training only supervoxel similarities and applying single linkage clustering produced less improvement.

## 1 Introduction

A clustering is defined as a partitioning of a set of elements into subsets called clusters. Roughly speaking, similar elements should belong to the same cluster and dissimilar ones to different clusters. In the traditional unsupervised formulation of clustering, the true membership of elements in clusters is completely unknown. Recently there has been interest in the supervised or semisupervised setting [5], in which true membership is known for some elements and can serve as training data. The goal is to learn a clustering algorithm that generalizes to new elements and new clusters. A convenient objective function for learning is the agreement between the output of the algorithm and the true clustering, for which the standard measurement is the Rand index [25].

Clustering is relevant for many application domains. One prominent example is image segmentation, the division of an image into clusters of pixels that correspond to distinct objects in the scene. Traditional approaches treated image segmentation as unsupervised clustering. However, it is becoming popular to utilize a supervised clustering approach in which a segmentation algorithm is trained on a set of images for which ground truth is known [23, 32]. The Rand index has become increasingly popular for evaluating the accuracy of image segmentation [34, 3, 13, 15, 35], and has recently been used as an objective function for supervised learning of this task [32].

This paper focuses on agglomerative algorithms for clustering, which iteratively merge pairs of clusters that maximize a similarity function. Equivalently, the merged pairs may be those that minimize

a distance or dissimilarity function, which is like a similarity function up to a change of sign. Speed is a chief advantage of agglomerative algorithms. The number of evaluations of the similarity function is polynomial in the number of elements to be clustered. In contrast, the popular approach of using a Markov random field to partition a graph with nodes that are the elements to be clustered, and edge weights given by their similarities, involves a computation that can be NP-hard [18].

Inefficient inference becomes even more costly for learning, which generally involves many iterations of inference. To deal with this problem, many researchers have developed learning methods for graphical models that depend on efficient *approximate* inference. However, once such approximations are introduced, many of the desirable theoretical properties of this framework no longer apply and performance in practice may be arbitrarily poor, as several authors have recently noted [36, 19, 8]. Here we avoid such issues by basing learning on agglomerative clustering, which is an efficient inference procedure in the first place.

We show that an agglomerative clustering algorithm can be regarded as a policy for a deterministic Markov decision process (DMDP) in which a state is a clustering, an action is a merging of two clusters, and the immediate reward is the change in the Rand index with respect to the ground truth clustering. In this formulation, the optimal action-value function turns out to be the optimal similarity function for agglomerative clustering. This DMDP formulation is helpful because it enables the application of ideas from reinforcement learning (RL) to find an approximation to the optimal similarity function.

Our formalism is generally applicable to any type of clustering, but is illustrated with a specific application to segmenting images by clustering superpixels. These are defined as groups of pixels from an oversegmentation produced by some other algorithm [27]. Recent research has shown that agglomerating superpixels using a hand-designed similarity function can improve segmentation accuracy [3]. It is plausible that it would be even more powerful to learn the similarity function from training data. Here we apply our RL framework to accomplish this, yielding a new method called Learning Agglomeration of Superpixel Hierarchies (LASH). LASH works by iteratively updating an approximation to the optimal similarity function. It uses the current approximation to generate a sequence of clusterings, and then improves the approximation on all possible actions on these clusterings.

LASH is an instance of a strategy called on-policy control in RL. This strategy has seen many empirical successes, but the theoretical guarantees are rather limited. Furthermore, LASH is implemented here for simplicity using infinite temporal discounting, though it could be extended to the case of finite discounting. Therefore we empirically evaluated LASH on the problem of segmenting images of brain tissue from serial electron microscopy, which has recently attracted a great deal of interest [6, 15]. We find that LASH substantially improves upon state of the art convolutional network and random forest boundary-detection methods for this problem, reducing segmentation error (as measured by the Rand error) by $50\%$ as compared to the next best technique.

We also tried the simpler strategy of directly training superpixel similarities, and then applying single linkage clustering [2]. This produced less accurate test set segmentations than LASH.

## 2 Agglomerative clustering as reinforcement learning

A Markov decision process (MDP) is defined by a state $s$, a set of actions $A(s)$ at each state, a function $P(s, a, s')$ specifying the probability of the $s \to s'$ transition after taking action $a \in A(s)$, and a function $R(s, a, s')$ specifying the immediate reward. A policy $\pi$ is a map from states to actions, $a = \pi(s)$. The goal of reinforcement learning (RL) is to find a policy $\pi$ that maximizes the expected value of total reward.

Total reward is defined as the sum of immediate rewards $\sum_{t=0}^{T-1} R(s_t, a_t)$ up to some time horizon $T$. Alternatively, it is defined as the sum of discounted immediate rewards, $\sum_{t=0}^{\infty} \gamma^t R(s_t, a_t)$, where $0 \leq \gamma \leq 1$ is the discount factor. Many RL methods are based on finding an optimal action-value function $Q^*(s, a)$, which is defined as the sum of discounted rewards obtained by taking action $a$ at state $s$ and following the optimal policy thereafter. An optimal policy can be extracted from this function by $\pi^*(s) = \operatorname{argmax}_a Q^*(s, a)$.

We can define agglomerative clustering as an MDP. Its state $s$ is a clustering of a set of objects. For each pair of clusters in $s_t$, there is an action $a_t \in A(s_t)$ that merges them to yield the clustering $s_{t+1} = a_t(s_t)$. Since the merge action is deterministic, we have the special case of a deterministic MDP, rather than a stochastic one. To define the rewards of the MDP, we make use of the Rand index, a standard measure of agreement between two clusterings of the same set [25]. A clustering is equivalent to classifying all pairs of objects as belonging to the same cluster or different clusters. The Rand index $RI(s, s')$ is the fraction of object pairs on which the clusterings $s$ and $s'$ agree. Therefore, we can define the immediate reward of action $a$ as the resulting increase in the Rand index with respect to a ground truth clustering $s^*$, $R(s, a) = RI(a(s), s^*) - RI(s, s^*)$.

An agglomerative clustering algorithm is a policy of this MDP, and the optimal similarity function is given by the optimal action-value function $Q^*$. The sum of undiscounted immediate rewards "telescopes" to the simple result $\sum_{t=0}^{T-1} R(s_t, a_t) = RI(s_T, s^*) - RI(s_0, s^*)$ [21]. Therefore RL for a finite time horizon $T$ is equivalent to maximizing the Rand index $RI(s_T, s^*)$ of the clustering at time $T$.

We will focus on the simple case of infinite discounting ($\gamma = 0$). Then the optimal action-value function $Q^*(s, a)$ is equal to $R(s, a)$. In other words, $R(s, a)$ is the best similarity function. We know $R(s, a)$ exactly for the training data, but we would also like it to apply to data for which ground truth is unknown. Therefore we train a function approximator $Q_\theta$ so that $Q_\theta(s, a) \approx R(s, a)$ on the training data, and hope that it generalizes to the test data. The following procedure is a simple way of doing this.

1. Generate an initial sequence of clusterings $(s_1, \ldots, s_T)$ by using $R(s, a)$ as a similarity function: iterate $a_t = \mathrm{argmax}_a R(s_t, a)$ and $s_{t+1} = a_t(s_t)$, terminating when $\max_a R(s_t, a) \leq 0$.

2. Train the parameters $\theta$ so that $Q_\theta(s_t, a) \approx R(s_t, a)$ for all $s_t$ and for all $a \in A(s_t)$.

3. Generate a new sequence of clusterings by using $Q_\theta(s, a)$ as a similarity function: iterate $a_t = \mathrm{argmax}_a Q_\theta(s_t, a)$ and $s_{t+1} = a_t(s_t)$, terminating when $\max_a Q_\theta(s_t, a) \leq 0$.

4. Goto 2.

Here the clustering $s_1$ is the trivial one in which each element is its own cluster. (The termination of the clustering is equivalent to the continued selection of a "do-nothing" action that leaves the clustering the same, $s_{t+1} = s_t$.) This is an example of "on-policy" learning, because the function approximator $Q_\theta$ is trained on clusterings generated by using it as a policy. It makes intuitive sense to optimize $Q_\theta$ for the kinds of clusterings that it actually sees in practice, rather than for all possible clusterings. However, there is no theoretical guarantee that such on-policy learning will converge, since we are using a nonlinear function approximation. Guarantees only exist if the action-value function is represented by a lookup table or a linear approximation. Nevertheless, the nonlinear approach has achieved practical success in a number of problem domains. Later we will present empirical results supporting the effectiveness of on-policy learning in our application.

The assumption of infinite discounting removes a major challenge of RL, dealing with temporally delayed reward. Are we losing anything by this assumption? If our approximation to the action-value function were perfect, $Q_\theta(s, a) = R(s, a)$, then agglomerative clustering would amount to greedy maximization of the Rand index. It is straightforward to show that this yields the clustering that is the global maximum. In practice, the approximation will be imperfect, and extending the above procedure to finite discounting could be helpful.

## 3  Agglomerating superpixels for image segmentation

The introduction of the Berkeley segmentation database (BSD) provoked a renaissance of the boundary detection and segmentation literature. The creation of a ground-truth segmentation database enabled learning-driven methods for low-level boundary detection, which were found to outperform classic methods such as Canny's [23, 10]. Global and multi-scale features were added to improve performance even further [26, 22, 29], and recently learning methods have been developed that directly optimize measures of segmentation performance [32, 13].

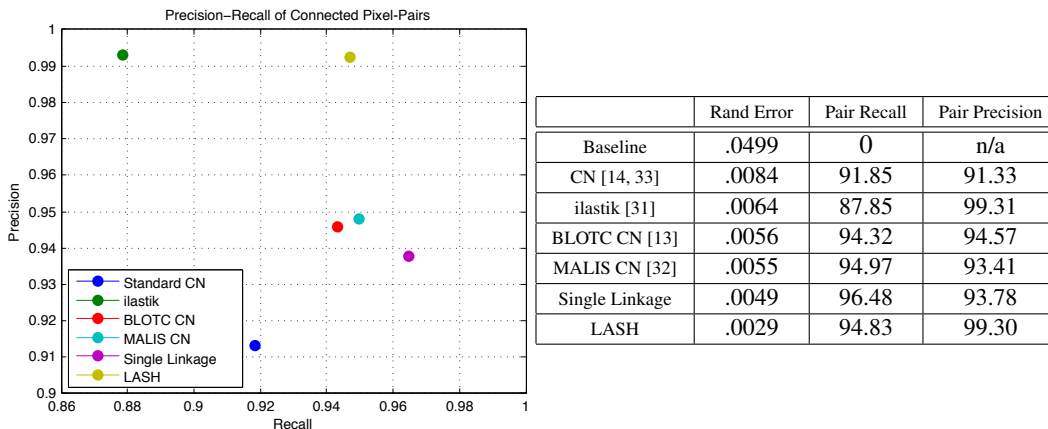

Figure 1: Performance comparison on a one megavoxel test set; parameters, such as the binarization threshold for the convolutional network (CN) affinity graphs, were determined based on the optimal value on the training set. CN's used a field of view of $16 \times 16 \times 16$, ilastik used a field of view of $23 \times 23 \times 23$, and LASH used a field of view of $50 \times 50 \times 50$. LASH leads to a substantial decrease in Rand error (1-Rand Index), and much higher connected pixel-pair precision at similar levels of recall as compared to other state of the art methods. The 'Connected pixel-pairs' curve measures the accuracy of connected pixel pairs pairs relative to ground truth. This measure corrects for the imbalance in the Rand error for segmentations in which most pixels are disconnected from one another, as in the case of EM reconstruction of dense brain wiring. For example, 'Trivial Baseline' above represents the trivial segmentation in which all pixels are disconnected from one another, and achieves relatively low Rand error but of course zero connected-pair recall.

However, boundary detectors alone have so far failed to produce segmentations that rival human levels of accuracy. Therefore many recent studies use boundary detectors to generate an oversegmentation of the image into fragments, and then attempt to cluster the "superpixels" . This approach has been shown to improve the accuracy of segmenting natural images [3, 30].

A similar approach [2, 1, 17, 35, 16] has also been employed to segment 3d nanoscale images from serial electron microscopy [11, 9]. In principle, it should be possible to map the connections between neurons by analyzing these images [20, 12, 28]. Since this analysis is highly laborious, it would be desirable to have automated computer algorithms for doing so [15]. First, each synapse must be identified. Second, the "wires" of the brain, its axons and dendrites, must be traced, i.e., segmented. If these two tasks are solved, it is then possible to establish which pairs of neurons are connected by synapses.

For our experiments, images of rabbit retina inner plexiform layer were acquired using Serial Block Face Scanning Electron Microscopy (SBF-SEM) [9, 4]. The tissue was specially stained to enhance cell boundaries while suppressing contrast from intracellular structures (e.g., mitochondria). The image volume was acquired at $22 \times 22 \times 25$ nm resolution, yielding a nearly isotropic 3d dataset with excellent slice-to-slice registration. Two training sets were created by human tracing and proofreading of subsets of the 3d image. The training sets were augmented with their eight 3d orthogonal rotations and reflections to yield 16 training images that contained roughly 80 megavoxels of labeled training data. A separate one megavoxel labeled test set was used to evaluate algorithm performance.

## 3.1 Boundary Detectors

For comparison purposes, as well as to provide supervoxels for LASH, we tested several state of the art boundary detection algorithms on the data. A convolutional network (CN) was trained to produce affinity graphs that can be segmented using connected components or watershed [14, 33]. We also trained CNs using MALIS and BLOTC, which are recently proposed machine learning algorithms that optimize true metrics of segmentation performance. MALIS directly optimizes the Rand index [32]. BLOTC, originally introduced for 2d boundary maps and here generalized to 3d affinity graphs,

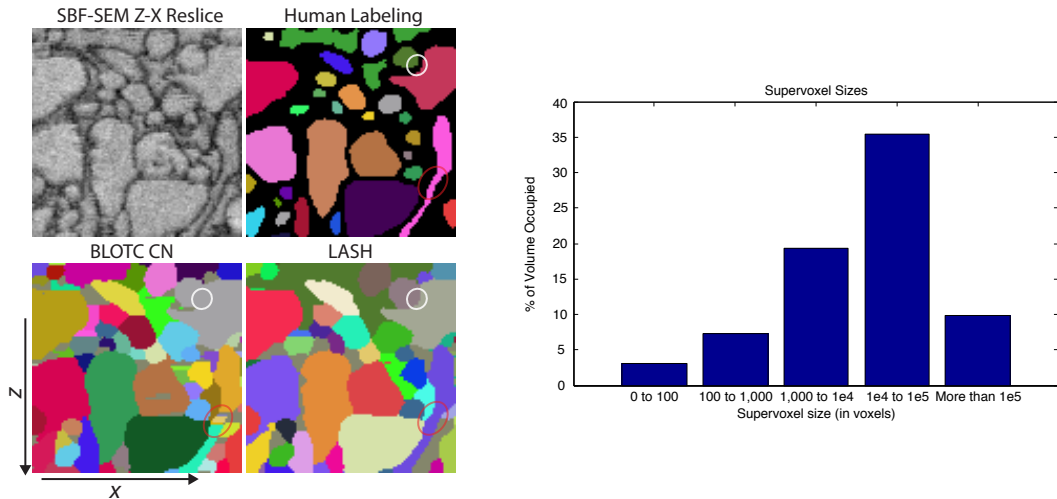

Figure 2: (Left) Visual comparison of output from a state of the boundary detector, BLOTC CN [13], and Learning to Agglomerate Superpixel Hierarchies (LASH). Image and segmentations are from a *Z-X* axis re-sectioning of the $100 \times 100 \times 100$ voxel test set. Segmentations were performed in 3d though only a single 2d $100 \times 100$ reslice is shown here. White circle shows an example location in which BLOTC CN merged two separate objects due to weak staining in an adjacent image slice; orange ellipse shows an example location in which BLOTC CN split up a single thin object. LASH avoids both of these errors. (Right) Distribution of supervoxel sizes, as measured by percentage of image volume occupied by specific size ranges of supervoxels.

optimizes 'warping error,' a measure of topological disagreement derived from concepts introduced in digital topology [13].

Finally, we trained 'ilastik,' a random-forest based boundary detector [31]. Unlike the CNs, which operated on the raw image and learned features as part of the training process, ilastik uses a pre-defined set of image features that represented low-level image structure such as intensity gradients and texture. The CNs used a field of view of $16 \times 16 \times 16$ voxels to make decisions about any particular image location, while ilastik used features from a field of view of up to $23 \times 23 \times 23$ voxels.

To generate segmentations of the test set, we found connected components of the thresholded boundary detector output, and then performed marker-based watershed to grow out these regions until they touched. Figure 1 shows the Rand index attained by the CNs and ilastik. Here we convert the index into an error measure by subtracting it from 1. Segmentation performance is sensitive to the threshold used to binarize boundary detector output, so we used the threshold that minimized Rand error on the training set.

### 3.2 Supervoxel Agglomeration

Supervoxels were generated from BLOTC convolutional network output, using connected components applied at a high threshold (0.9) to avoid undersegmented regions (in the test set, there was only one supervoxel in the initial oversegmentation which contained more than one ground truth region). Regions were then grown out using marker-based watershed. The size of the supervoxels varied considerably, but the majority of the image volume was assigned to supervoxels larger than $1,000$ voxels in size (as shown in Figure 3).

For each pair of neighboring supervoxels, we computed a 138 dimensional feature vector, as described in the Appendix. This was used as input to the learned similarity function $Q_\theta$, which we represented by a decision-tree boosting classifier [7]. We followed the procedure given in Section 2, but with two modifications. First the examples used in each training iteration were collected by segmenting all the images in the training set, not only a single image. Second, $Q_\theta$ was trained to approximate $H(R(s_t, a))$ rather than $R(s_t, a)$, where $H$ is the Heaviside step function and the log-loss was optimized. This was done because our function approximator was suitable for classification, but

some other approximator suitable for regression could also be used. The loop in the procedure of Section 2 was terminated when training error stopped decreasing by a significant amount, after 3 cycles. Then the learned similarity function was applied to agglomerate supervoxels in the test set to yield the results in Figure 1. The agglomeration terminated after around 5000 steps.

The results show substantial decrease in Rand error compared to state of the art techniques (MALIS and BLOTC CN). A potential sublety in interpreting these results is the small absolute values of the Rand error for *all* of these techniques. The Rand error is defined as the probability of classifying pairs of voxels as belonging to the same or different clusters. This classification task is highly imbalanced, because the vast majority of voxel pairs belong to different ground truth clusters. Hence even a completely trivial segmentation in which every voxel is its own cluster can achieve fairly low Rand error (Figure 1). Precision and recall are better quantifications of performance at imbalanced classification[23]. Figure 1 shows that LASH achieves much higher precision at similar recall. For the task of segmenting neurons in EM images, high precision is especially important as false positives can lead to false positive neuron-pair connections.

Visual comparison of segmentation performance is shown in Figure 2. LASH avoids both split and merge errors that result from segmenting BLOTC CN output. BLOTC CN in turn was previously shown to outperform other techniques such as Boosted Edge Learning, multi-scale normalized cut, and gPb-OWT-UCM [13].

### 3.3 Naive training of the similarity function on superpixel pairs

In the conventional algorithms for agglomerative clustering, the similarity $S(A, B)$ of two clusters $A$ and $B$ can be reduced to the similarities $S(x, y)$ of elements $x \in A$ and $y \in B$. For example, single linkage clustering assumes that $S(A, B) = \max_{x \in A, y \in B} S(x, y)$. The maximum operation is replaced by the minimum or average in other common algorithms. LASH does not impose any such constraint of reducibility on the similarity function. Consequently, LASH must truly compute new similarities after each agglomerative step. In contrast, conventional algorithms can start by computing the matrix of similarities between the elements to be clustered, and all further similarities between clusters follow from trivial computations.

Therefore another method of learning agglomerative clustering is to train a similarity function on pairs of superpixels only, and then apply a standard agglomerative algorithm such as single linkage clustering. This has been previously been done for images from serial electron microscopy [2]. (Note that single linkage clustering is equivalent to creating a graph in which nodes are superpixels and edge weights are their similarities, and then finding the connected components of the thresholded graph.) As shown in Figure 1, clustering superpixels in this way improves upon boundary detection algorithms. However, the improvement is substantially less than achieved by LASH.

## Discussion

Why did LASH achieve better accuracy than other approaches? One might argue that the comparison is unfair, because the CNs and ilastik detected boundaries using a field of view considerably smaller than that used in the LASH features (up to $50 \times 50 \times 50$ for the SVF feature computation). If these competing methods were allowed to use the same context, perhaps their accuracy would improve dramatically. This is possible, but training time would also increase dramatically. Training a CN with MALIS or BLOTC on 80 megavoxels of training data with a $16^3$ field of view already takes on the order of a week, using an optimized GPU implementation [24]. Adding the additional layers to the CN required to achieve a field of view of $50^3$ might require months of additional training.[1] In contrast, the entire LASH training process is completed within roughly one day. This can be attributed to the efficiency gains associated with computations on supervoxels rather than voxels. In short, LASH is more accurate because it is efficient enough to utilize more image context in its computations.

Why does LASH outperform the naive method of directly training superpixel similarities used in single linkage clustering? The naive method uses the same amount of image context. In this case,

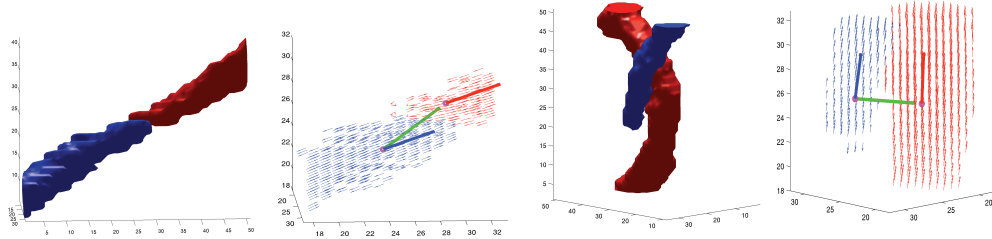

Figure 3: Example of SVF feature computation. Blue and red are two different supervoxels. Left panel shows rendering of the objects, right panel shows smoothed vector fields (thin arrows), along with chosen center-of-mass orientation vectors (thick blue/red lines) and line connecting the two center of masses (thick green line). The angle between the thick blue/red and green lines is used as a feature during LASH.

LASH is probably superior because it trains the similarities by optimizing the clustering that they actually yield. The naive method resembles LASH, but with the modification that the action-value function is trained only for the actions possible on the clustering $s_1$ rather than on the entire sequence of clusterings (see Step 2 of the procedure in Section 2).

We have conceptualized LASH in the framework of reinforcement learning. Previous work has applied reinforcement learning to other structured prediction problems [21]. An additional closely related approach to structured prediction is SEARN, introduced by Daume et al [8]. As in our approach, SEARN uses a single classifier repeatedly on a (structured) input to iteratively solve an inference problem. The major difference between our approach and theirs is the way the classifier is trained. In paticular, SEARN begins with a manually specified policy (given by ground truth or heuristics) and then iteratively *degrades* the policy as a classifier is trained and 'replaces' the initial policy. In our approach, the initial policy may exhibit poor performance (i.e., for random initial $\theta$), and then improves through training.

We have implemented LASH with infinite discounting of future rewards, but extending to finite discounting might produce better results. Generalizing the action space to include splitting of clusters as well as agglomeration might also be advantageous. Finally, the objective function optimized by learning might be tailored to better reflect more task-specific criteria, such as the number of locations that human might have to correct ('proofread') to yield an error-free segmentation by semiautomated means. These directions will be explored in future work.

## Appendix

### Features of supervoxel pairs used by the similarity function

The similarity function that we trained with LASH required as input a set of features for each supervoxel pair that might be merged. For each supervoxel pair, we first computed a 'decision point,' defined as the midpoint of the shortest line that connects any two points of the supervoxels. From this decision point, we computed several types of features that encodes information about the underlying affinity graph as well the shape of the supervoxel objects near the decision point: (1) size of each supervoxel in the pair, (2) distance between the two supervoxels, (3) analog affinity value of the graph edge at which the two supervoxels would merge if grown out using watershed, and the distance from the decision point to this edge, (4) 'Smoothed Vector Field' (SVF), a novel shape feature described below, computed at various spatial scales (maximum $50 \times 50 \times 50$). This feature measures the orientation of each supervoxel near the decision point.

Finally, for each supervoxel in the pair we also included the above features for the closest 4 other decision points that involve that supervoxel. Overall, this feature set yielded a 138 dimensional feature vector for each supervoxel pair.

The smoothed vector field (SVF) shape feature attempts to determine the orientation of a supervoxel near some specific location (e.g., the decision point used in reference to some other supervoxel).

The main challenge in computing such an orientation is dealing with high-frequency noise and irregularities in the precise shape of the supervoxel. We developed a novel approach that deals with this issue by smoothing a vector field derived from image moments. For a binary 3d image, SVF is computed in the following manner:

1. A spherical mask of radius 5 is selected around each image location $I_{x,y,z}$, and $\vec{v}_{x,y,z}$ is then computed as the largest eigenvector of the $3 \times 3$ second order image moment matrix for that window.

2. The vector field is smoothed via 3 iterations of 'ising-like' interactions among nearest neighbor vector fields: $\vec{v}_{x,y,z} \leftarrow f\left(\sum_{i=x-1}^{x+1}\sum_{j=y-1}^{y+1}\sum_{k=z-1}^{z+1}\vec{v}_{i,j,k}\right)$, where $f$ represents a (non-linear) renormalization such that the magnitude of each vector remains 1.

3. The smoothed vector at the center of mass of the supervoxel is used to compute angular orientation of the supervoxel (see Figure 3).

## Footnotes

[1]Using a much larger field of view with a CN will likely require new architectures that incorporate multi-scale capabilities.

# References

[1] B. Andres, J. H. Kappes, U. Köthe, C. Schnörr, and F. A. Hamprecht. An empirical comparison of inference algorithms for graphical models with higher order factors using opengm. In M. Goesele, S. Roth, A. Kuijper, B. Schiele, and K. Schindler, editors, *Pattern Recognition*, volume 6376 of *Lecture Notes in Computer Science*, pages 353–362. Springer, 2010.

[2] B. Andres, U. Koethe, M. Helmstaedter, W. Denk, and F. Hamprecht. Segmentation of SBFSEM volume data of neural tissue by hierarchical classification. In *Proceedings of the 30th DAGM symposium on Pattern Recognition*, pages 142–152. Springer, 2008.

[3] P. Arbelaez, M. Maire, C. Fowlkes, and J. Malik. From contours to regions: An empirical evaluation. *Computer Vision and Pattern Recognition, IEEE Computer Society Conference on*, 0:2294–2301, 2009.

[4] K. L. Briggman and W. Denk. Towards neural circuit reconstruction with volume electron microscopy techniques. *Current opinion in neurobiology*, 16(5):562–570, 2006.

[5] O. Chapelle, B. Schlkopf, and A. Zien. Semi-supervised learning. *The MIT Press*, page 528, 2010.

[6] D. Chklovskii, S. Vitaladevuni, and L. Scheffer. Semi-automated reconstruction of neural circuits using electron microscopy. *Current Opinion in Neurobiology*, 2010.

[7] M. Collins, R. Schapire, and Y. Singer. Logistic regression, AdaBoost and Bregman distances. *Machine Learning*, 48(1):253–285, 2002.

[8] H. Daumé, III, J. Langford, and D. Marcu. Search-based structured prediction. *Machine Learning*, 75:297–325, 2009. 10.1007/s10994-009-5106-x.

[9] W. Denk and H. Horstmann. Serial block-face scanning electron microscopy to reconstruct three-dimensional tissue nanostructure. *PLoS Biol*, 2(11):e329, 2004.

[10] P. Dollar, Z. Tu, and S. Belongie. Supervised learning of edges and object boundaries. *Computer Vision and Pattern Recognition, IEEE Computer Society Conference on*, 2:1964–1971, 2006.

[11] K. J. Hayworth, N. Kasthuri, R. Schalek, and J. W. Lichtman. Automating the collection of ultrathin serial sections for large volume TEM reconstructions. *Microscopy and Microanalysis*, 12(S02):86–87, 2006.

[12] M. Helmstaedter, K. L. Briggman, and W. Denk. 3D structural imaging of the brain with photons and electrons. *Current Opinion in Neurobiology*, 18(6):633–641, 2008.

[13] V. Jain, B. Bollmann, M. Richardson, D. Berger, M. Helmstaedter, K. Briggman, W. Denk, J. Bowden, J. Mendenhall, W. Abraham, K. Harris, N. Kasthuri, K. Hayworth, R. Schalek, J. Tapia, J. Lichtman, and H. Seung. Boundary Learning by Optimization with Topological Constraints. In *Computer Vision and Pattern Recognition, IEEE Computer Society Conference on*, 2010.

[14] V. Jain, J. F. Murray, F. Roth, S. C. Turaga, V. Zhigulin, K. L. Briggman, M. N. Helmstaedter, W. Denk, and H. S. Seung. Supervised learning of image restoration with convolutional networks. *Computer Vision, IEEE International Conference on*, 0:1–8, 2007.

[15] V. Jain, H. Seung, and S. Turaga. Machines that learn to segment images: a crucial technology for connectomics. *Current opinion in neurobiology*, 2010.

[16] E. Jurrus, R. Whitaker, B. W. Jones, R. Marc, and T. Tasdizen. An optimal-path approach for neural circuit reconstruction. In *Biomedical Imaging: From Nano to Macro, 2008. ISBI 2008. 5th IEEE International Symposium on*, pages 1609–1612, May 2008.

[17] V. Kaynig, T. Fuchs, and J. M. Buhmann. Neuron geometry extraction by perceptual grouping in sstem images. In *Computer Vision and Pattern Recognition, IEEE Computer Society Conference on*, 2010.

[18] V. Kolmogorov and R. Zabih. What energy functions can be minimizedvia graph cuts? *IEEE Transactions on Pattern Analysis and Machine Intelligence*, pages 147–159, 2004.

[19] A. Kulesza, F. Pereira, et al. Structured learning with approximate inference. *Advances in neural information processing systems*, 20, 2007.

[20] J. W. Lichtman and J. R. Sanes. Ome sweet ome: what can the genome tell us about the connectome? *Curr. Opin. Neurobiol.*, 18(3):346–353, Jun 2008.

[21] F. Maes, L. Denoyer, and P. Gallinari. Structured prediction with reinforcement learning. *Machine learning*, 77(2):271–301, 2009.

[22] M. Maire, P. Arbeláez, C. Fowlkes, and J. Malik. Using contours to detect and localize junctions in natural images. In *Computer Vision and Pattern Recognition, 2008. CVPR 2008. IEEE Conference on*, pages 1–8. IEEE, 2008.

[23] D. R. Martin, C. C. Fowlkes, and J. Malik. Learning to detect natural image boundaries using local brightness, color, and texture cues. *IEEE Trans. Patt. Anal. Mach. Intell.*, pages 530–549, 2004.

[24] J. Mutch, U. Knoblich, and T. Poggio. CNS: a GPU-based framework for simulating cortically-organized networks. Technical report, Massachussetts Institute of Technology, 2010.

[25] W. M. Rand. Objective criteria for the evaluation of clustering methods. *Journal of the American Statistical association*, 66(336):846–850, 1971.

[26] X. Ren. Multi-scale improves boundary detection in natural images. In *Proceedings of the 10th European Conference on Computer Vision: Part III*, pages 533–545. Springer-Verlag, Springer, 2008.

[27] X. Ren and J. Malik. Learning a Classification Model for Segmentation. In *Proceedings of the Ninth IEEE International Conference on Computer Vision-Volume 2*, page 10. IEEE Computer Society, 2003.

[28] H. Seung. Reading the Book of Memory: Sparse Sampling versus Dense Mapping of Connectomes. *Neuron*, 62(1):17–29, 2009.

[29] E. Sharon, A. Brandt, and R. Basri. Fast multiscale image segmentation. In *Computer Vision and Pattern Recognition, 2000. Proceedings. IEEE Conference on*, volume 1, pages 70–77. IEEE, 2000.

[30] E. Sharon, M. Galun, D. Sharon, R. Basri, and A. Brandt. Hierarchy and adaptivity in segmenting visual scenes. *Nature*, 442(7104):810–813, 2006.

[31] C. Sommer, C. Straehle, U. Köthe, and F. A. Hamprecht. "ilastik: Interactive learning and segmentation toolkit". In *8th IEEE International Symposium on Biomedical Imaging (ISBI 2011), in press*, 2011.

[32] S. C. Turaga, K. L. Briggman, M. Helmstaedter, W. Denk, and H. S. Seung. Maximin affinity learning of image segmentation. In *NIPS*, 2009.

[33] S. C. Turaga, J. F. Murray, V. Jain, F. Roth, M. Helmstaedter, K. Briggman, W. Denk, and H. S. Seung. Convolutional networks can learn to generate affinity graphs for image segmentation. *Neural Computation*, 22(2):511–538, 2010.

[34] R. Unnikrishnan, C. Pantofaru, and M. Hebert. Toward objective evaluation of image segmentation algorithms. *IEEE Transactions on Pattern Analysis and Machine Intelligence*, 29(6):929, 2007.

[35] S. N. Vitaladevuni and R. Basri. Co-clustering of image segments using convex optimization applied to EM neuronal reconstruction. In *Computer Vision and Pattern Recognition, IEEE Computer Society Conference on*, 2010.

[36] M. Wainwright. Estimating the wrong graphical model: Benefits in the computation-limited setting. *The Journal of Machine Learning Research*, 7:1829–1859, 2006.

